# Human Rademacher Complexity

**Xiaojin Zhu**[1]**, Timothy T. Rogers**[2]**, Bryan R. Gibson**[1]
Department of {[1]Computer Sciences, [2]Psychology}
University of Wisconsin-Madison. Madison, WI 15213
`jerryzhu@cs.wisc.edu, ttrogers@wisc.edu, bgibson@cs.wisc.edu`

## Abstract

We propose to use Rademacher complexity, originally developed in computational learning theory, as a measure of human learning capacity. Rademacher complexity measures a learner's ability to fit random labels, and can be used to bound the learner's true error based on the observed training sample error. We first review the definition of Rademacher complexity and its generalization bound. We then describe a "learning the noise" procedure to experimentally measure human Rademacher complexities. The results from empirical studies showed that: (i) human Rademacher complexity can be successfully measured, (ii) the complexity depends on the domain and training sample size in intuitive ways, (iii) human learning respects the generalization bounds, (iv) the bounds can be useful in predicting the danger of overfitting in human learning. Finally, we discuss the potential applications of human Rademacher complexity in cognitive science.

## 1 Introduction

Many problems in cognitive psychology arise from questions of *capacity*. How much information can human beings hold in mind and deploy in simple memory tasks [19, 15, 6]? What kinds of functions can humans easily acquire when learning to classify items [29, 7], and do they have biases for learning some functions over others[10]? Is there a single domain-general answer to these questions, or is the answer domain-specific [28]? How do human beings avoid over-fitting learning examples when acquiring knowledge that allows them to generalize [20]? Such questions are central to a variety of research in cognitive psychology, but only recently have they begun to be placed on a formal mathematical footing [7, 9, 5].

Machine learning offers a variety of formal approaches to measuring the capacity of a learning system, with concepts such as Vapnik-Chervonenkis (VC) dimension [27, 25, 12] and Rademacher complexity [1, 13, 24]. Based on these notions of capacity, one can quantify the generalization performance of a classifier, and the danger of over-fitting, by bounding its future test error using its observed training sample error. In this paper, we show how one such concept–Rademacher complexity–can be measured in humans, based on their performance in a "learning the noise" procedure. We chose Rademacher complexity (rather than the better-known VC dimension) because it is particularly amenable to experimental studies, as discussed in Section 5. We assess whether human capacity varies depending on the nature of the materials to be categorized, and empirically test whether human generalization behavior respects the error bounds in a variety of categorization tasks. The results validate Rademacher complexity as a meaningful measure of human learning capacity, and provide a new perspective on the human tendency to overfit training data in category learning tasks. We note that our aim is not to develop a new formal approach to complexity, but rather to show how a well-studied formal measure can be computed for human beings.

## 2 Rademacher Complexity

**Background and definitions.** Let $\mathcal{X}$ be a domain of interest, which in psychology corresponds to a stimulus space. For example, $\mathcal{X}$ could be an infinite set of images parametrized by some continuous parameters, a finite set of words, etc.. We will use $x \in \mathcal{X}$ to denote an instance (e.g., an image or a word) from the domain; precisely how $x$ is represented is immaterial. We assume there is an underlying marginal distribution $P_X$ on $\mathcal{X}$, such that $x$ is sampled with probability $P_X(x)$ during training and testing. For example, $P_X$ can be uniform on the set of words.

Let $f : \mathcal{X} \mapsto \mathbb{R}$ be a real-valued function. This corresponds to a hypothesis that predicts the label of any instance in the domain. The label can be a continuous value for regression, or $\{-1, 1\}$ for binary classification. Let $\mathcal{F}$ be the set of such functions, or the hypothesis space, that we consider. For example, in machine learning $\mathcal{F}$ may be the set of linear classifiers. In the present work, we will take $\mathcal{F}$ to be the (possibly infinite) set of hypotheses from $\mathcal{X}$ to binary classes $\{-1, 1\}$ that humans can come up with.

Rademacher complexity (see for example [1]) measures the capacity of the hypothesis space $\mathcal{F}$ by how easy it is for $\mathcal{F}$ to fit random noise. Consider a sample of $n$ instances: $x_1, \ldots, x_n$ drawn i.i.d. from $P_X$. Now generate $n$ random numbers $\sigma_1, \ldots, \sigma_n$, each taking value -1 or 1 with equal probability. For a given function $f \in \mathcal{F}$, its fit to the random numbers is defined as $|\sum_{i=1}^{n} \sigma_i f(x_i)|$. This is easier to understand when $f$ produces -1, 1 binary labels. In this case, the $\sigma$'s can be thought of as random labels, and $\{(x_i, \sigma_i)\}_{i=1}^{n}$ as a training sample. The fit measures how $f$'s predictions match the random labels on the training sample: if $f$ perfectly predicts the $\sigma$'s, or completely the opposite by flipping the classes, then the fit is maximized at $n$; if $f$'s predictions are orthogonal to the $\sigma$'s, the fit is minimized at 0.

The fit of a set of functions $\mathcal{F}$ is defined as $\sup_{f \in \mathcal{F}} |\sum_{i=1}^{n} \sigma_i f(x_i)|$. That is, we are fitting the particular training sample by finding the hypothesis in $\mathcal{F}$ with the best fit. If $\mathcal{F}$ is rich, it will be easier to find a hypothesis $f \in \mathcal{F}$ that matches the random labels, and its fit will be large. On the other hand, if $\mathcal{F}$ is simple (e.g., in the extreme containing only one function $f$), it is unlikely that $f(x_i)$ will match $\sigma_i$, and its fit will be close to zero.

Finally, recall that $\{(x_i, \sigma_i)\}_{i=1}^{n}$ is a particular random training sample. If, for every random training sample of size $n$, there always exists some $f \in \mathcal{F}$ (which may be different each time) that matches it, then $\mathcal{F}$ is very good at fitting random noise. This also means that $\mathcal{F}$ is prone to overfitting, whose very definition is to learn noise. This is captured by taking the expectation over training samples:

**Definition 1** (Rademacher Complexity). *For a set of real-valued functions $\mathcal{F}$ with domain $\mathcal{X}$, a distribution $P_X$ on $\mathcal{X}$, and a size $n$, the Rademacher complexity $R(\mathcal{F}, \mathcal{X}, P_X, n)$ is*

$$R(\mathcal{F}, \mathcal{X}, P_X, n) = \mathbb{E}_{\boldsymbol{x}\boldsymbol{\sigma}} \left[ \sup_{f \in \mathcal{F}} \left| \frac{2}{n} \sum_{i=1}^{n} \sigma_i f(x_i) \right| \right], \tag{1}$$

*where the expectation is over $\boldsymbol{x} = x_1, \ldots, x_n \overset{iid}{\sim} P_X$, and $\boldsymbol{\sigma} = \sigma_1, \ldots, \sigma_n \overset{iid}{\sim} \text{Bernoulli}(\frac{1}{2}, \frac{1}{2})$ with values $\pm 1$.*

Rademacher complexity depends on the hypothesis space $\mathcal{F}$, the domain $\mathcal{X}$, the distribution on the domain $P_X$, as well as the training sample size $n$. The size $n$ is relevant because for a fixed $\mathcal{F}$, it will be increasingly difficult to fit random noise as $n$ gets larger. On the other hand, it is worth noting that Rademacher complexity is independent of any future classification tasks. For example, in Section 4 we will discuss two different tasks on the same $\mathcal{X}$ (set of words): classifying a word by its emotional valence, or by its length. These two tasks will share the same Rademacher complexity. In general, the value of Rademacher complexity will depend on the range of $\mathcal{F}$. In the special case when $\mathcal{F}$ is a set of functions mapping $x$ to $\{-1, 1\}$, $R(\mathcal{F}, \mathcal{X}, P_X, n)$ is between 0 and 2.

A particularly important property of Rademacher complexity is that it can be estimated from random samples. Let $\{(x_i^{(1)}, \sigma_i^{(1)})\}_{i=1}^{n}, \ldots, \{(x_i^{(m)}, \sigma_i^{(m)})\}_{i=1}^{n}$ be $m$ random samples of size $n$ each. In Section 3, these will correspond to $m$ different subjects. The following theorem is an extension of Theorem 11 in [1]. The proof follows from McDiarmid's inequality [16].

**Theorem 1.** *Let $\mathcal{F}$ be a set of functions mapping to $[-1, 1]$. For any integers $n, m$,*

$$P\left\{\left|R(\mathcal{F}, \mathcal{X}, P_X, n) - \frac{1}{m}\sum_{j=1}^{m}\sup_{f \in \mathcal{F}}\left|\frac{2}{n}\sum_{i=1}^{n}\sigma_i^{(j)}f(x_i^{(j)})\right|\right| \geq \epsilon\right\} \leq 2\exp\left(-\frac{\epsilon^2 nm}{8}\right) \quad (2)$$

Theorem 1 allows us to estimate the expectation in (1) with random samples, which is of practical importance. It remains to compute the supremum in (1). In Section 3, we will discuss our procedure to approximate the supremum in the case of human learning.

**Generalization Error Bound.** We state a generalization error bound by Bartlett and Mendelson (Theorem 5 in [1]) as an important application of Rademacher complexity. Consider any binary classification task of predicting a label in $\mathcal{Y} = \{-1, 1\}$ for $x \in \mathcal{X}$. For example, the label $y$ could be the emotional valence (positive=1 vs. negative=-1) of a word $x$. In general, a binary classification task is characterized by a joint distribution $P_{XY}$ on $\mathcal{X} \times \{-1, 1\}$. Training data and future test data consist of instance-label pairs $(x, y) \overset{iid}{\sim} P_{XY}$. Let $\mathcal{F}$ be a set of binary classifiers that map $\mathcal{X}$ to $\{-1, 1\}$. For $f \in \mathcal{F}$, let $(y \neq f(x))$ be an indicator function which is 1 if $y \neq f(x)$, and 0 otherwise. On a training sample $\{(x_i, y_i)\}_{i=1}^{n}$ of size $n$, the observed training sample error of $f$ is $\hat{e}(f) = \frac{1}{n}\sum_{i=1}^{n}(y_i \neq f(x_i))$. The more interesting quantity is the true error of $f$, i.e., how well $f$ can generalize to future test data: $e(f) = \mathbb{E}_{(x,y)\overset{iid}{\sim}P_{XY}}[(y \neq f(x))]$. Rademacher complexity allows us to bound the true error using training sample error as follows.

**Theorem 2.** *(Bartlett and Mendelson) Let $\mathcal{F}$ be a set of functions mapping $\mathcal{X}$ to $\{-1, 1\}$. Let $P_{XY}$ be a probability distribution on $\mathcal{X} \times \{-1, 1\}$ with marginal distribution $P_X$ on $\mathcal{X}$. Let $\{(x_i, y_i)\}_{i=1}^{n} \overset{iid}{\sim} P_{XY}$ be a training sample of size $n$. For any $\delta > 0$, with probability at least $1 - \delta$, every function $f \in F$ satisfies*

$$e(f) - \hat{e}(f) \leq \frac{R(\mathcal{F}, \mathcal{X}, P_X, n)}{2} + \sqrt{\frac{\ln(1/\delta)}{2n}}. \quad (3)$$

The probability $1 - \delta$ is over random draws of the training sample. That is, if one draws a large number of training samples of size $n$ each, then (3) is expected to hold on $1 - \delta$ fraction of those samples. The bound has two factors, one from the Rademacher complexity and the other from the confidence parameter $\delta$ and training sample size $n$. When the bound is tight, training sample error is a good indicator of true error, and we can be confident that overfitting is unlikely. A tight bound requires the Rademacher complexity to be close to zero. On the other hand, if the Rademacher complexity is large, or $n$ is too small, or the requested confidence $1 - \delta$ is overly stringent, the bound can be loose. In that case, there is a danger of overfitting. We will demonstrate this generalization error bound on four different human classification tasks in Section 4.

## 3 Measuring Human Rademacher Complexity by Learning the Noise

Our aim is to measure the Rademacher complexity of the human learning system for a given stimulus space $\mathcal{X}$, distribution of instances $P_X$, and sample-size $n$. By "human learning system," we mean the set of binary classification functions that an average human subject can come up with on the domain $\mathcal{X}$, under the experiment conditions described below. We will denote this set of functions $\mathcal{F}$ with $H_a$, that is, "average human."

We make two assumptions. The first is the assumption of universality [2]: every individual has the same $H_a$. It allows us to pool subjects together. This assumption can be loosened in the future. For instance, $\mathcal{F}$ could be defined as the set of functions that *a particular individual or group* can employ in the learning task, such as a given age-group, education level, or other special population. A second assumption is required to compute the supremum on $H_a$. Obviously, we cannot measure and compare the performance of every single function contained in $H_a$. Instead, we assume that, when making their classification judgments, participants use the best function at their disposal–so that the errors they make when tested on the training instances reflect the error generated by the best-performing function in $H_a$, thus providing a direct measure of the supremum. In essence, the assumption is that participants are doing their best to perform the task.

With the two assumptions above, we can compute human Rademacher complexity for a given stimulus domain $\mathcal{X}$, distribution $P_X$, and sample size $n$, by assessing how well human participants are able to learn randomly-assigned labels. Each participant is presented with a training sample $\{(x_i, \sigma_i)\}_{i=1}^n$ where the $\sigma$'s are random $\pm 1$ labels, and asked to learn the instance-label mapping. The subject is not told that the labels are random. We assume that the subject will search within $H_a$ for the best hypothesis ("rule"), which is the one that minimizes training error: $f^* = \arg\max_{f \in H_a} \sum_{i=1}^n \sigma_i f(x_i) = \arg\min_{f \in H_a} \hat{e}(f)$. We do not directly observe $f^*$. Later, we ask the subject to classify the same training instances $\{x_i\}_{i=1}^n$ using what she has learned. Her classification labels will be $f^*(x_1), \ldots, f^*(x_n)$, which we do observe. We then approximate the supremum as follows: $\sup_{f \in H_a} \left| \frac{2}{n} \sum_{i=1}^n \sigma_i f(x_i) \right| \approx \left| \frac{2}{n} \sum_{i=1}^n \sigma_i f^*(x_i) \right|$. For the measured Rademacher complexity to reflect actual learning capacity on the set $H_a$, it is important to prevent participants from simply doing rote learning. With these considerations, we propose the following procedure to estimate human Rademacher complexity.

**Procedure.** Given domain $\mathcal{X}$, distribution $P_X$, training sample size $n$, and number of subjects $m$, we generate $m$ random samples of size $n$ each: $\{(x_i^{(1)}, \sigma_i^{(1)})\}_{i=1}^n, \ldots, \{(x_i^{(m)}, \sigma_i^{(m)})\}_{i=1}^n$, where $x_i^{(j)} \overset{iid}{\sim} P_X$ and $\sigma_i^{(j)} \overset{iid}{\sim}$ Bernoulli$(\frac{1}{2}, \frac{1}{2})$ with value $\pm 1$, for $j = 1 \ldots m$. The procedure is paper-and-pencil based, and consists of three steps:

**Step 1.** Participant $j$ is shown a printed sheet with the training sample $\{(x_i^{(j)}, \sigma_i^{(j)})\}_{i=1}^n$, where each instance $x_i^{(j)}$ is paired with its random label $\sigma_i^{(j)}$ (shown as "A" and "B" instead of -1,1 for convenience). the participant is informed that there are only two categories; the order does not matter; they have three minutes to study the sheet; and later they will be asked to use what they have learned to categorize more instances into "A" or "B".

**Step 2.** After three minutes the sheet is taken away. To prevent active maintenance of training items in working memory the participant performs a filler task consisting of ten two-digit addition/subtraction questions.

**Step 3.** The participant is given another sheet with the same training instances $\{x_i^{(j)}\}_{i=1}^n$ but no labels. The order of the $n$ instances is randomized and different from step 1. The participant is not told that they are the same training instances, and is asked to categorize each instance as "A" or "B" and is encouraged to guess if necessary. There is no time limit.

Let $f^{(j)}(x_1^{(j)}), \ldots, f^{(j)}(x_n^{(j)})$ be subject $j$'s answers (encoded as $\pm 1$). We estimate $R(H_a, \mathcal{X}, P_X, n)$ by $\frac{1}{m} \sum_{j=1}^m \left| \frac{2}{n} \sum_{i=1}^n \sigma_i^{(j)} f^{(j)}(x_i^{(j)}) \right|$. We also conduct a post-experiment interview where the subject reports any insights or hypotheses they may have on the categories.

**Materials** To instantiate the general procedure, one needs to specify the domain $\mathcal{X}$ and an associated marginal distribution $P_X$. For simplicity, in all our experiments $P_X$ is the uniform distribution over the corresponding domain. We conducted experiments on example domains. They are not of specific interest in themselves but nicely illustrate many interesting properties of human Rademacher complexity: **(1) The "Shape" Domain**. $\mathcal{X}$ consists of 321 computer-generated 3D shapes [3]. The shapes are parametrized by a real number $x \in [0, 1]$, such that small $x$ produces spiky shapes, while large $x$ produces smooth ones. A few instances and their parameters are shown in Figure 1(a). It is important to note that this internal structure is unnecessary to the definition or measurement of Rademacher complexity *per se*. However, in Section 4 we will define some classification tasks that utilize this internal structure. Participants have little existing knowledge about this domain. **(2) The "Word" Domain**. $\mathcal{X}$ consists of 321 English words. We start with the Wisconsin Perceptual Attribute Ratings Database [18], which contains words rated by 350 undergraduates for their emotional valence. We sort the words by their emotion valence, and take the 161 most positive and the 160 most negative ones for use in the study. A few instances and their emotion ratings are shown in Figure 1(b). Participants have rich knowledge about this domain. The size of the domain for shapes and words was matched to facilitate comparison.

**Participants** were 80 undergraduate students, participating for partial course credit. They were divided evenly into eight groups. Each group of $m = 10$ subjects worked on a unique combination of the Shape or the Word domain, and training sample size $n$ in 5, 10, 20, or 40, using the procedure defined previously.

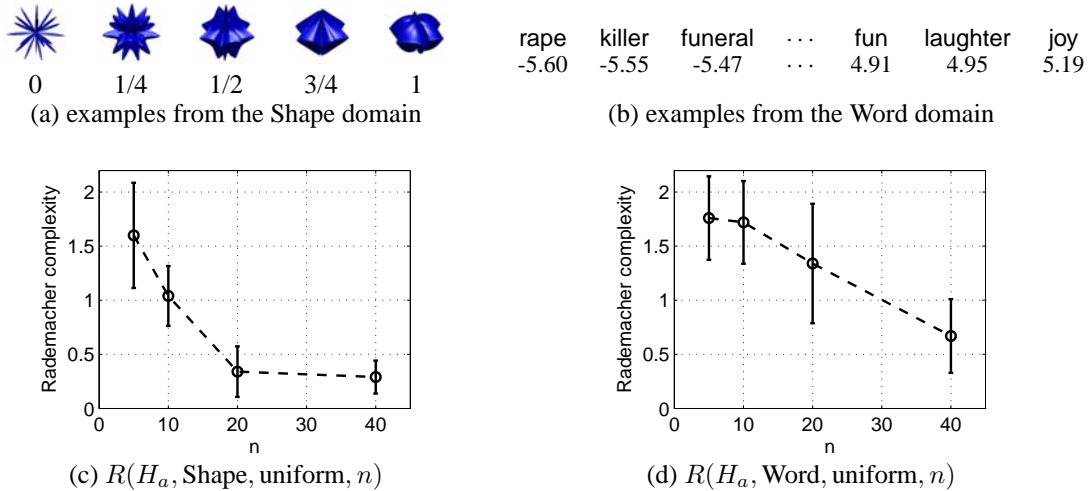

|  | rape | killer | funeral | $\cdots$ | fun | laughter | joy |
|---|---|---|---|---|---|---|---|
|  | -5.60 | -5.55 | -5.47 | $\cdots$ | 4.91 | 4.95 | 5.19 |

(a) examples from the Shape domain          (b) examples from the Word domain

(c) $R(H_a, \text{Shape}, \text{uniform}, n)$          (d) $R(H_a, \text{Word}, \text{uniform}, n)$

Figure 1: Human Rademacher complexity on the "Shape" and "Word" domains.

**Results**. Figures 1(c,d) show the measured human Rademacher complexities on the domains $\mathcal{X}$=Shape and Word respectively, with distribution $P_X$=uniform, and with different training sample sizes $n$. The error bars are 95% confidence intervals. Several interesting observations can be made from the data:

*Observation 1: human Rademacher complexities in both domains decrease as $n$ increases*. This is anticipated, as it should be harder to learn a larger number of random labels. Indeed, when $n = 5$, our interviews show that, in both domains, 9 out of 10 participants offered some spurious rules of the random labels. For example, one subject thought the shape categories were determined by whether the shape "faces" downward; another thought the word categories indicated whether the word contains the letter T. Such beliefs, though helpful in learning the particular training samples, amount to over-fitting the noise. In contrast, when $n = 40$, about half the participants indicated that they believed the labels to be random, as spurious "rules" are more difficult to find.

*Observation 2: human Rademacher complexities are significantly higher in the Word domain than in the Shape domain*, for $n = 10, 20, 40$ respectively (t-tests, $p < 0.05$). The higher complexity indicates that, for the same sample sizes, participants are better able to find spurious explanations of the training data for the Words than for the Shapes. Two distinct strategies were apparent in the Word domain interviews: (i) Some participants created mnemonics. For example, one subject received the training sample (grenade, B), (skull, A), (conflict, A), (meadow, B), (queen, B), and came up with the following story: "a queen was sitting in a meadow and then a grenade was thrown (B = before), then this started a conflict ending in bodies & skulls (A = after)." (ii) Other participants came up with idiosyncratic, but often imperfect, rules. For instance, whether the item "tastes good," "relates to motel service," or "physical vs. abstract." We speculate that human Rademacher complexities on other domains can be drastically different too, reflecting the richness of the participant's pre-existing knowledge about the domain.

*Observation 3: many of these human Rademacher complexities are relatively large*. This means that under those $\mathcal{X}, P_X, n$, humans have a large capacity to learn arbitrary labels, and so will be more prone to overfit on real (i.e., non-random) tasks. We will present human generalization experiments in Section 4. It is also interesting to note that both Rademacher complexities at $n = 5$ are less than 2: under our procedure, participants are not perfect at remembering the labels of merely five instances.

## 4  Bounding Human Generalization Errors

We reiterate the interpretation of human Rademacher complexity for psychology. It does not predict $\hat{e}$ (how well humans perform when training for a given task). Instead, Theorem 2 bounds $e - \hat{e}$, the "amount of overfitting" (sometimes also called "instability") when the subject switches from training to testing. A tight (close to 0) bound guarantees no severe overfitting: humans' future

Table 1: Human learning performance abides by the generalization error bounds.

| condition | ID | $\hat{e}$ | bound $e$ | $e$ | condition | ID | $\hat{e}$ | bound $e$ | $e$ |
|---|---|---|---|---|---|---|---|---|---|
| Shape-+ n=5 | 81 | 0.00 | 1.35 | 0.05 | WordEmotion n=5 | 101 | 0.00 | 1.43 | 0.58 |
| | 82 | 0.00 | 1.35 | 0.22 | | 102 | 0.00 | 1.43 | 0.46 |
| | 83 | 0.00 | 1.35 | 0.10 | | 103 | 0.00 | 1.43 | 0.04 |
| | 84 | 0.00 | 1.35 | 0.09 | | 104 | 0.00 | 1.43 | 0.03 |
| | 85 | 0.00 | 1.35 | 0.07 | | 105 | 0.00 | 1.43 | 0.31 |
| Shape-+ n=40 | 86 | 0.05 | 0.39 | 0.04 | WordEmotion n=40 | 106 | 0.70 | 1.23 | 0.65 |
| | 87 | 0.03 | 0.36 | 0.14 | | 107 | 0.00 | 0.53 | 0.04 |
| | 88 | 0.03 | 0.36 | 0.03 | | 108 | 0.00 | 0.53 | 0.00 |
| | 89 | 0.00 | 0.34 | 0.04 | | 109 | 0.62 | 1.15 | 0.53 |
| | 90 | 0.00 | 0.34 | 0.01 | | 110 | 0.00 | 0.53 | 0.05 |
| Shape-+- n=5 | 91 | 0.00 | 1.35 | 0.23 | WordLength n=5 | 111 | 0.00 | 1.43 | 0.46 |
| | 92 | 0.00 | 1.35 | 0.27 | | 112 | 0.00 | 1.43 | 0.69 |
| | 93 | 0.00 | 1.35 | 0.21 | | 113 | 0.00 | 1.43 | 0.55 |
| | 94 | 0.00 | 1.35 | 0.40 | | 114 | 0.00 | 1.43 | 0.26 |
| | 95 | 0.20 | 1.55 | 0.18 | | 115 | 0.00 | 1.43 | 0.57 |
| Shape-+- n=40 | 96 | 0.12 | 0.46 | 0.16 | WordLength n=40 | 116 | 0.12 | 0.65 | 0.51 |
| | 97 | 0.32 | 0.66 | 0.50 | | 117 | 0.45 | 0.98 | 0.55 |
| | 98 | 0.15 | 0.49 | 0.08 | | 118 | 0.00 | 0.53 | 0.00 |
| | 99 | 0.15 | 0.49 | 0.11 | | 119 | 0.15 | 0.68 | 0.29 |
| | 100 | 0.03 | 0.36 | 0.10 | | 120 | 0.15 | 0.68 | 0.37 |

test performance $e$ will be close to their training performance $\hat{e}$. This does not mean they will do *well*: $\hat{e}$ could be large and thus $e$ is similarly large. A loose bound, in contrast, is a warning sign for overfitting: good training performance (small $\hat{e}$) may not reflect learning of the correct categorization rule, and so does not entail good performance on future samples (i.e., $e$ can be much larger than $\hat{e}$). We now present four non-random category-learning tasks to illustrate these points.

**Materials.** We consider four very different binary classification tasks to assess whether Theorem 2 holds for all of them. The tasks are: **(1) Shape-+**: Recall the Shape domain is parametrized by $x \in [0, 1]$. The task has a linear decision boundary at $x = 0.5$, i.e., $P(y = 1|x) = 0$ if $x < 0.5$, and 1 if $x \geq 0.5$. It is well-known that people can easily learn such boundaries, so this is a fairly easy task on the domain. **(2) Shape-+-**: This task is also on the Shape domain, but with a nonlinear decision boundary. The negative class is on both ends while the positive class is in the middle: $P(y = 1|x) = 0$ if $x \in [0, 0.25) \cup (0.75, 1]$, and 1 if $x \in [0.25, 0.75]$. Prior research suggests that people have difficulty learning nonlinearly separable categories [28, 7], so this is a harder task. Note, however, that the two shape tasks share the same Rademacher complexity, and therefore have the same bound for the same $n$. **(3) WordEmotion**: This task is on the Word domain. $P(y = 1|x) = 0$ if word $x$ has a negative emotion rating in the Wisconsin Perceptual Attribute Ratings Database, and $P(y = 1|x) = 1$ otherwise. **(4) WordLength**: $P(y = 1|x) = 0$ if word $x$ has 5 or less letters, and $P(y = 1|x) = 1$ otherwise. The two word tasks are drastically different in that one focuses on semantics and the other on orthography, but they share the same Rademacher complexity and thus the same bound (for the same $n$), because the underlying domain is the same.

**Procedure.** The procedure is identical to that in Section 3 except for two things: (i) Instead of random labels $\sigma$, we sample labels $y \overset{iid}{\sim} P(y|x)$ appropriate for each task. (ii) In step 3, in addition to the training instances $\{x_i^{(j)}\}_{i=1}^n$, the $j^{th}$ subject is also given 100 test instances $\{x_i^{(j)}\}_{i=n+1}^{n+100}$, sampled from $P_X$. The order of the training and test instances is randomized. The true test labels $y$ are sampled from $P(y|x)$. We compute the participant's training sample error as $\hat{e}(f^{(j)}) = 1/n \sum_{i=1}^n \left( y_i \neq f^{(j)}(x_i^{(j)}) \right)$, and estimate her generalization error as $e(f^{(j)}) = 1/100 \sum_{i=n+1}^{n+100} \left( y_i \neq f^{(j)}(x_i^{(j)}) \right)$.

**Participants** were 40 additional students, randomly divided into 8 groups of five each. Each group worked on one of the four tasks, with training sample size $n=5$ or 40.

**Results.** We present the performance of individual participants in Table 1: $\hat{e}$ is the observed training error for that subject, "bound $e$" is the 95% confidence (i.e., $\delta = 0.05$) bound on test error:

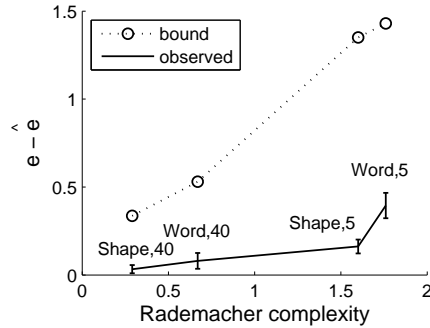

Figure 2: Human Rademacher complexity predicts the trend of overfitting.

$\hat{e}(f) + R(\mathcal{F}, \mathcal{X}, P_X, n)/2 + \sqrt{\ln(1/\delta)/2n}$, and $e$ is the observed test error. We also present the aggregated results across subjects and tasks in Figure 2, comparing the bound on $e - \hat{e}$ (the "amount of overfitting," RHS of (3)) vs. the observed $e - \hat{e}$, as the underlying Rademacher complexity varies. We make two more observations:

*Observation 4: Theorem 2 holds for every participant.* Table 1 provides empirical support that our application of computational learning theory to human learning is viable. In fact, for our choice of $\delta = 0.05$, Theorem 2 allows the bound to fail on about two (5% of 40) participants – which did not happen. Of course, some of the "bound $e$" are vacuous (greater than 1) as it is well-known that bounds in computational learning theory are not always tight [14], but others are reasonably tight (e.g., on Shape-+ with $n = 40$).

*Observation 5: the larger the Rademacher complexity, the worse the actual amount of overfitting $e - \hat{e}$.* Figure 2 shows that as $R$ increases, $e - \hat{e}$ increases (solid line; error bar $\pm$standard error; averaged over the two different tasks with the same domain and $n$, as noted in the graph). The bound on $e - \hat{e}$ (dotted line; RHS of (3)) has the same trend, although, being loose, it is higher up. This seems to be true regardless of the classification task. For example, the Word domain and $n = 5$ has a large Rademacher complexity 1.76, and both task WordLength and task WordEmotion severely overfit: In task WordLength with $n = 5$, all subjects had zero training error but had large test error, suggesting that their good performance on the training items reflects overfitting. Accordingly, the explanations offered during the post-test interviews for this group spuriously fit the training items but did not reflect the true categorization rule. Subject 111 thought that the class decision indicated "things you can go inside," while subject 114 thought the class indicated an odd or even number of syllables. Similarly, on task WordEmotion with $n = 5$, three out of five subjects overfit the training items. Subject 102 received the training items (daylight, 1), (hospital, -1), (termite, -1), (envy, -1), (scream, -1), and concluded that class 1 is "anything related to omitting[sic] light," and proceeded to classify the test items as such.

## 5   Discussions and Future Work

Is our study on *memory* or *learning*? This distinction is not necessarily relevant here, as we adopt an abstract perspective which analyzes the human system as a black box that produces labels, and both learning and memory contribute to the process being executed in that black box. We do have evidence from post-interviews that Figure 1 does not merely reflect list-length effects from memory studies: (i) participants treated the study as a category-learning and not a memory task – they were not told that the training and test items are the same when we estimate $R$; (ii) the memory load was identical in the shape and the word domains, yet the curves differ markedly; (iii) participants were able to articulate the "rules" they were using to categorize the items.

Much recent research has explored the relationship between the statistical complexity of some categorization task and the ease with which humans learn the task [7, 5, 9, 11]. Rademacher complexity is different: it indexes not the complexity of the $\mathcal{X} \mapsto \mathcal{Y}$ categorization task, but the sophistication of the learner in domain $\mathcal{X}$ (note $\mathcal{Y}$ does not appear in Rademacher complexity). Greater complexity indicates, not a more difficult categorization task, but a greater tendency to overfit sparse data.

On the other hand, our definition of Rademacher complexity depends only on the domain, distribution, and sample size. In human learning, other factors also contribute to learnability, such as the instructions, motivation, time to study, and should probably be incorporated into the complexity.

Human Rademacher complexity has interesting connections to other concepts. The VC-dimension [27, 25, 12] is another capacity measure. Let $\{x_1, \ldots, x_m\} \subseteq \mathcal{X}$ be a subset of the domain. Let $(f(x_1), \ldots, f(x_m))$ be a $\pm 1$-valued vector which is the classifications made by some $f \in \mathcal{F}$. If $\mathcal{F}$ is rich enough such that its members can produce all $2^m$ vectors: $\{(f(x_1), \ldots, f(x_m)) : f \in \mathcal{F}\} = \{-1, 1\}^m$, then we say that the subset is shattered by $\mathcal{F}$. The VC-dimension of $\mathcal{F}$ is the size of the largest subset that can be shattered by $\mathcal{F}$, or $\infty$ if $\mathcal{F}$ can shatter arbitrarily large subsets. Unfortunately, human VC-dimension seems difficult to measure experimentally: First, shattering requires validating an exponential ($2^m$) number of classifications on a given subset. Second, to determine that the VC-dimension is $m$, one needs to show that no subset of size $m + 1$ can be shattered. However, the number of such subsets can be infinite. A variant, "effective VC-dimension", may be empirically estimated from a training sample [26]. This is left for future research. Normalized Maximum Likelihood (NML) uses a similar complexity measure for a model class [21], the connection merits further study ([23], p.50).

Human Rademacher complexity might help to advance theories of human cognition in many ways. First, human Rademacher complexity can provide a means of testing computational models of human concept learning. Traditionally, such models are assessed by comparing their performance to human performance in terms of classification error. A new approach would be to derive or empirically estimate the Rademacher complexity of the computational models, and compare that to human Rademacher complexity. A good computational model should match humans in this regard.

Second, our procedure could be used to measure human Rademacher complexity in individuals or special populations, including typically and atypically-developing children and adults. Relating individual Rademacher complexity to standard measures of learning ability or IQ may shed light on the relationship between complexity, learning, and intelligence. Many IQ tests measure the participant's ability to generalize the pattern in words or images. Individuals with very high Rademacher complexity may actually perform worse by being "distracted" by other potential hypotheses.

Third, human Rademacher complexity may help explain the human tendency to discern patterns in random stimuli, such as the well-known Rorschach inkblot test, "illusory correlations" [4], or "false-memory" effect [22]. These effects may be viewed as spurious rule-fitting to (or generalization of) the observed data, and Human Rademacher complexity may quantify the possibility of observing such an effect.

Fourth, cognitive psychologists have long entertained an interest in characterizing the capacity of different mental processes such as, for instance, the capacity limitations of short-term memory [19, 6]. In this vein, our work suggests a different kind of metric for assessing the capacity of the human learning system.

Finally, human Rademacher complexity can help experimental psychologists to determine the propensity of overfitting in their stimulus materials. We have seen that human Rademacher complexity can be much higher in some domains (e.g. Word) than others (e.g. Shape). Our procedure could be used to measure the human Rademacher complexity of many standard concept-learning materials in cognitive science, such as the Greebles used by Tarr and colleagues [8] and the circle-and-line stimuli of McKinley & Nosofsky [17].

**Acknowledgment**: We thank the reviewers for their helpful comments. XZ thanks Michael Coen for discussions that lead to the realization of the difficulties in measuring human VC dimension. This work is supported in part by AFOSR grant FA9550-09-1-0313 and the Wisconsin Alumni Research Foundation.

# References

[1] P. L. Bartlett and S. Mendelson. Rademacher and Gaussian complexities: risk bounds and structural results. *Journal of Machine Learning Research*, 3:463–482, 2002.

[2] A. Caramazza and M. McCloskey. The case for single-patient studies. *Cognitive Neuropsychology*, 5(5):517–527, 1988.

[3] R. Castro, C. Kalish, R. Nowak, R. Qian, T. Rogers, and X. Zhu. Human active learning. In *Advances in Neural Information Processing Systems (NIPS) 22*. 2008.

[4]  L. J. Chapman. Illusory correlation in observational report. *Journal of Verbal Learning and Verbal Behavior*, 6:151–155, 1967.

[5]  N. Chater and P. Vitanyi. Simplicity: A unifying principle in cognitive science? *Trends in Cognitive Science*, 7(1):19–22, 2003.

[6]  N. Cowan. The magical number 4 in short-term memory: A reconsideration of mental storage capacity. *Behavioral and Brain Sciences*, 24:87–185, 2000.

[7]  J. Feldman. Minimization of boolean complexity in human concept learning. *Nature*, 407:630–633, 2000.

[8]  I. Gauthier and M. Tarr. Becoming a "greeble" expert: Exploring mechanisms for face recognition. *Vision Research*, 37(12):1673–1682, 1998.

[9]  N. Goodman, J. B. Tenenbaum, J. Feldman, and T. L. Griffiths. A rational analysis of rule-based concept learning. *Cognitive Science*, 32(1):108–133, 2008.

[10]  T. L. Griffiths, B. R. Christian, and M. L. Kalish. Using category structures to test iterated learning as a method for identifying inductive biases. *Cognitive Science*, 32:68–107, 2008.

[11]  T. L. Griffiths and J. B. Tenenbaum. From mere coincidences to meaningful discoveries. *Cognition*, 103(2):180–226, 2007.

[12]  M. J. Kearns and U. V. Vazirani. *An Introduction to Computational Learning Theory*. MIT Press, 1994.

[13]  V. I. Koltchinskii and D. Panchenko. Rademacher processes and bounding the risk of function learning. In E. Gine, D. Mason, and J. Wellner, editors, *High Dimensional Probability II*, pages 443–459. MIT Press, 2000.

[14]  J. Langford. Tutorial on practical prediction theory for classification. *Journal of Machine Learning Research*, 6:273–306, 2005.

[15]  R. L. Lewis. Interference in short-term memory: The magical number two (or three) in sentence processing. *Journal of Psycholinguistic Research*, 25(1):93–115, 1996.

[16]  C. McDiarmid. On the method of bounded differences. In *Surveys in Combinatorics 1989*, pages 148–188. Cambridge University Press, 1989.

[17]  S. C. McKinley and R. M. Nosofsky. Selective attention and the formation of linear decision boundaries. *Journal of Experimental Psychology: Human Perception & Performance*, 22(2):294–317, 1996.

[18]  D. Medler, A. Arnoldussen, J. Binder, and M. Seidenberg. The Wisconsin perceptual attribute ratings database, 2005. http://www.neuro.mcw.edu/ratings/.

[19]  G. Miller. The magical number seven plus or minus two: Some limits on our capacity for processing information. *Psychological Review*, 63(2):81–97, 1956.

[20]  R. C. O'Reilly and J. L. McClelland. Hippocampal conjunctive encoding, storage, and recall: Avoiding a tradeoff. *Hippocampus*, 4:661–682, 1994.

[21]  J. Rissanen. Strong optimality of the normalized ML models as universal codes and information in data. *IEEE Transaction on Information Theory*, 47(5):17121717, 2001.

[22]  H. L. Roediger and K. B. McDermott. Creating false memories: Remembering words not presented in lists. *Journal of Experimental Psychology: Learning, Memory and Cognition*, 21(4):803–814, 1995.

[23]  T. Roos. *Statistical and Information-Theoretic Methods for Data Analysis*. PhD thesis, Department of Computer Science, University of Helsinki, 2007.

[24]  J. Shawe-Taylor and N. Cristianini. *Kernel Methods for Pattern Analysis*. Cambridge University Press, New York, NY, USA, 2004.

[25]  V. Vapnik. *Statistical Learning Theory*. Wiley-Interscience, 1998.

[26]  V. Vapnik, E. Levin, and Y. Le Cun. Measuring the VC-dimension of a learning machine. *Neural Computation*, 6:851–876, 1994.

[27]  V. N. Vapnik and A. Y. Chervonenkis. On the uniform convergence of relative frequencies of events to their probabilities. *Theory of Probability and its Applications*, 16(2):264–280, 1971.

[28]  W. D. Wattenmaker. Knowledge structures and linear separability: Integrating information in object and social categorization. *Cognitive Psychology*, 28(3):274–328, 1995.

[29]  W. D. Wattenmaker, G. I. Dewey, T. D. Murphy, and D. L. Medin. Linear separability and concept learning: Context, relational properties, and concept naturalness. *Cognitive Psychology*, 18(2):158–194, 1986.

